# Competitive On-Line Linear Regression

**V. Vovk**
Department of Computer Science
Royal Holloway, University of London
Egham, Surrey TW20 0EX, UK
`vovk@dcs.rhbnc.ac.uk`

## Abstract

We apply a general algorithm for merging prediction strategies (the Aggregating Algorithm) to the problem of linear regression with the square loss; our main assumption is that the response variable is bounded. It turns out that for this particular problem the Aggregating Algorithm resembles, but is slightly different from, the well-known ridge estimation procedure. From general results about the Aggregating Algorithm we deduce a guaranteed bound on the difference between our algorithm's performance and the best, in some sense, linear regression function's performance. We show that the AA attains the optimal constant in our bound, whereas the constant attained by the ridge regression procedure in general can be 4 times worse.

## 1  INTRODUCTION

The usual approach to regression problems is to assume that the data are generated by some stochastic mechanism and make some, typically very restrictive, assumptions about that stochastic mechanism. In recent years, however, a different approach to this kind of problems was developed (see, e.g., DeSantis et al. [2], Littlestone and Warmuth [7]): in our context, that approach sets the goal of finding an on-line algorithm that performs not much worse than the best regression function found off-line; in other words, it replaces the usual statistical analyses by the competitive analysis of on-line algorithms.

DeSantis et al. [2] performed a competitive analysis of the Bayesian merging scheme for the log-loss prediction game; later Littlestone and Warmuth [7] and Vovk [10] introduced an on-line algorithm (called the Weighted Majority Algorithm by the

former authors) for the simple binary prediction game. These two algorithms (the Bayesian merging scheme and the Weighted Majority Algorithm) are special cases of the Aggregating Algorithm (AA) proposed in [9, 11]. The AA is a member of a wide family of algorithms called "multiplicative weight" or "exponential weight" algorithms.

Closer to the topic of this paper, Cesa-Bianchi et al. [1] performed a competitive analysis, under the square loss, of the standard Gradient Descent Algorithm and Kivinen and Warmuth [6] complemented it by a competitive analysis of a modification of the Gradient Descent, which they call the Exponentiated Gradient Algorithm. The bounds obtained in [1, 6] are of the following type: at every trial $T$,

$$L_T \leq cL_T^* + O(1), \tag{1}$$

where $L_T$ is the loss (over the first $T$ trials) of the on-line algorithm, $L_T^*$ is the loss of the best (by trial $T$) linear regression function, and $c$ is a constant, $c > 1$; specifically, $c = 2$ for the Gradient Descent and $c = 3$ for the Exponentiated Gradient. These bounds hold under the following assumptions: for the Gradient Descent, it is assumed that the $L_2$ norm of the weights and of all data items are bounded by constant 1; for the Exponentiated Gradient, that the $L_1$ norm of the weights and the $L_\infty$ norm of all data items are bounded by 1.

In many interesting cases bound (1) is weak. For example, suppose that our comparison class contains a "true" regression function, but its values are corrupted by an i.i.d. noise. Then, under reasonable assumptions about the noise, $L_T^*$ will grow linearly in $T$, and inequality (1) will only bound the difference $L_T - L_T^*$ by a linear function of $T$. (Though in other situations bound (1) can be better than our bound (2), see below. For example, in the case of the Exponentiated Gradient, the $O(1)$ in (1) depends on the number of parameters $n$ logarithmically whereas our bound depends on $n$ linearly.)

In this paper we will apply the AA to the problem of linear regression. The AA has been proven to be optimal in some simple cases [5, 11], so we can also expect good performance in the problem of linear regression. The following is a typical result that can be obtained using the AA: Learner has a strategy which ensures that always

$$L_T \leq L_T^* + n\ln(T+1) + 1 \tag{2}$$

($n$ is the number of predictor variables). It is interesting that the assumptions for the last inequality are weaker than those for both the Gradient Descent and Exponentiated Gradient: we only assume that the $L_2$ norm of the weights and the $L_\infty$ norm of all data items are bounded by constant 1 (these assumptions will be further relaxed later on). The norms $L_2$ and $L_\infty$ are not dual, which casts doubt on the accepted intuition that the weights and data items should be measured by dual norms (such as $L_1$–$L_\infty$ or $L_2$–$L_2$).

Notice that the logarithmic term $n\ln(T+1)$ of (2) is similar to the term $\frac{n}{2}\ln T$ occurring in the analysis of the log-loss game and its generalizations, in particular in Wallace's theory of minimum message length, Rissanen's theory of stochastic complexity, minimax regret analysis. In the case $n = 1$ and $x_t = 1$, $\forall t$, inequality (2) differs from Freund's [4] Theorem 4 only in the additive constant. In this paper we will see another manifestation of a phenomenon noticed by Freund [4]: for some important problems, the adversarial bounds of on-line competitive learning theory

are only a tiny amount worse than the average-case bounds for some stochastic strategies for Nature.

A weaker variant of inequality (2) can be deduced from Foster's [3] Theorem 1 (if we additionally assume that the response variable take only two values, $-1$ or $1$): Foster's result implies

$$L_T \leq L_T^* + 8n \ln(2n(T+1)) + 8$$

(a multiple of 4 arises from replacing Foster's set $\{0,1\}$ of possible values of the response variable by our $\{-1,1\}$; we also replaced Foster's $d$ by $2n$: to span our set of possible weights we need $2n$ Foster's predictors).

Inequality (2) is also similar to Yamanishi's [12] result; in that paper, he considers a more general framework than ours but does not attempt to find optimal constants.

## 2   ALGORITHM

We consider the following protocol of interaction between Learner and Nature:

> FOR $t = 1, 2, \ldots$
> > Nature chooses $x_t \in \mathbb{R}^n$
> > Learner chooses prediction $p_t \in \mathbb{R}$
> > Nature chooses $y_t \in [-Y, Y]$
> END FOR.

This is a "perfect-information" protocol: either player can see the other player's moves. The parameters of our protocol are: a fixed positive number $n$ (the dimensionality of our regression problem) and an upper bound $Y > 0$ on the value $y_t$ returned by Nature. It is important, however, that our algorithm for playing this game (on the part of Learner) *does not need to know* $Y$.

We will only give a description of our regression algorithm; its derivation from the general AA will be given in the future full version of this paper. (It is usually a nontrivial task to represent the AA in a computationally efficient form, and the case of on-line linear regression is not an exception.) Fix $n$ and $a > 0$. The algorithm is as follows:

> $A := aI;\ b := 0$
> FOR TRIAL $t = 1, 2, \ldots$:
> > read new $x_t \in \mathbb{R}^n$
> > $A := A + x_t x_t'$
> > output prediction $p_t := b' A^{-1} x_t$
> > read new $y_t \in \mathbb{R}$
> > $b := b + y_t x_t$
> END FOR.

In this description, $A$ is an $n \times n$ matrix (which is always symmetrical and positive definite), $b \in \mathbb{R}^n$, $I$ is the unit $n \times n$ matrix, and $0$ is the all-0 vector.

The naive implementation of this algorithm would require $O(n^3)$ arithmetic operations at every trial, but the standard recursive technique allows us to spend only $O(n^2)$ arithmetic operations per trial. This is still not as good as for the Gradient Descent Algorithm and Exponentiated Gradient Algorithm (they require only $O(n)$

operations per trial); we seem to have a trade-off between the quality of bounds on predictive performance and computational efficiency. In the rest of the paper "AA" will mean the algorithm described in the previous paragraph (which is the Aggregating Algorithm applied to a particular uncountable pool of experts with a particular Gaussian prior).

## 3   BOUNDS

In this section we state, without proof, results describing the predictive performance of our algorithm. Our comparison class consists of the linear functions $y_t = w \cdot x_t$, where $w \in \mathbb{R}^n$. We will call the possible weights $w$ "experts" (imagine that we have continuously many experts indexed by $w \in \mathbb{R}^n$; Expert $w$ always recommends prediction $w \cdot x_t$ to Learner). At every trial $t$ Expert $w$ and Learner suffer loss $(y_t - w \cdot x_t)^2$ and $(y_t - p_t)^2$, respectively. Our notation for the total loss suffered by Expert $w$ and Learner over the first $T$ trials will be

$$L_T(w) := \sum_{t=1}^{T}(y_t - w \cdot x_t)^2$$

and

$$L_T(\text{Learner}) := \sum_{t=1}^{T}(y_t - p_t)^2,$$

respectively.

For compact pools of experts (which, in our setting, corresponds to the set of possible weights $w$ being bounded and closed) it is usually possible to derive bounds (such as (2)) where the learner's loss is compared to the best expert's loss. In our case of non-compact pool, however, we need to give the learner a start on remote experts. Specifically, instead of comparing Learner's performance to $\inf_w L_T(w)$, we compare it to $\inf_w \left(L_T(w) + a\|w\|^2\right)$ (thus giving Learner a start of $a\|w\|^2$ on Expert $w$), where $a > 0$ is a constant reflecting our prior expectations about the "complexity" $\|w\| := \sqrt{\sum_{i=1}^{n} w_i^2}$ of successful experts.

This idea of giving a start to experts allows us to prove stronger results; e.g., the following elaboration of (2) holds:

$$L_T(\text{Learner}) \le \inf_w \left(L_T(w) + \|w\|^2\right) + n\ln(T+1) \qquad (3)$$

(this inequality still assumes that $\|x_t\|_\infty \le 1$ for all $t$ but $w$ is unbounded).

Our notation for the transpose of matrix $A$ will be $A'$; as usual, vectors are identified with one-column matrices.

**Theorem 1** *For any fixed $n$, Learner has a strategy which ensures that always*

$$
\begin{aligned}
L_T(\text{Learner}) &\le \inf_w \left(L_T(w) + a\|w\|^2\right) + Y^2 \ln \det \left(I + \frac{1}{a}\sum_{t=1}^{T} x_t x_t'\right) \\
&\le \inf_w \left(L_T(w) + a\|w\|^2\right) + Y^2 \sum_{i=1}^{n} \ln \left(1 + \frac{1}{a}\sum_{t=1}^{T} x_{t,i}^2\right).
\end{aligned}
$$

*If, in addition, $\|x_t\|_\infty \leq X$, $\forall t$,*

$$L_T(\text{Learner}) \leq \inf_w \left( L_T(w) + a\|w\|^2 \right) + nY^2 \ln \left( \frac{TX^2}{a} + 1 \right). \tag{4}$$

The last inequality of this theorem implies inequality (3): it suffices to put $X = Y = a = 1$.

The term

$$\ln \det \left( I + \frac{1}{a} \sum_{t=1}^T x_t x_t' \right)$$

in Theorem 1 might be difficult to interpret. Notice that it can be rewritten as

$$n \ln T + \ln \det \left( \frac{1}{T} I + \frac{1}{a} \text{cov}(X_1, \ldots, X_n) \right),$$

where $\text{cov}(X_1, \ldots, X_n)$ is the empirical covariance matrix of the predictor variables (in other words, $\text{cov}(X_1, \ldots, X_n)$ is the covariance matrix of the random vector which takes the values $x_1, \ldots, x_T$ with equal probability $\frac{1}{T}$). We can see that this term is typically close to $n \ln T$.

Using standard transformations, it is easy to deduce from Theorem 1, e.g., the following results (for simplicity we assume $n = 1$ and $x_t, y_t \in [-1, 1]$, $\forall t$):

- if the pool of experts consists of all polynomials of degree $d$, Learner has a strategy guaranteeing

$$L_T(\text{Learner}) \leq \inf_w \left( L_T(w) + \|w\|^2 \right) + (d+1) \ln(T+1).$$

- if the pool of experts consists of all splines of degree $d$ with $k$ nodes (chosen *a priori*), Learner has a strategy guaranteeing

$$L_T(\text{Learner}) \leq \inf_w \left( L_T(w) + \|w\|^2 \right) + (d+k+1) \ln(T+1).$$

The following theorem shows that the constant $n$ in inequality (4) cannot be improved.

**Theorem 2** *Fix $n$ (the number of attributes) and $Y$ (the upper bound on $|y_t|$). For any $\epsilon > 0$ there exist a constant $C$ and a stochastic strategy for Nature such that $\|x_t\|_\infty = 1$ and $|y_t| = Y$, for all $t$, and, for any stochastic strategy for Learner,*

$$\mathbf{E} \left( L_T(\text{Learner}) - \inf_{w : \|w\| \leq Y} L_T(w) \right) \geq (n - \epsilon) Y^2 \ln T - C, \quad \forall T.$$

## 4   COMPARISONS

It is easy to see that the ridge regression procedure sometimes gives results that are not sensible in our framework where $y_t \in [-Y, Y]$ and the goal is to compete

against the best linear regression function. For example, suppose $n = 1$, $Y = 1$, and Nature generates outcomes $(x_t, y_t)$, $t = 1, 2, \ldots$, where

$$a \ll x_1 \ll x_2 \ll \ldots, \quad y_t = \begin{cases} 1, & \text{if } t \text{ odd}, \\ -1, & \text{if } t \text{ even}. \end{cases}$$

At trial $t = 2, 3, \ldots$ the ridge regression procedure (more accurately, its natural modification which truncates its predictions to $[-1, 1]$) will give prediction $p_t = y_{t-1}$ equal to the previous response, and so will suffer a loss of about $4T$ over $T$ trials. On the other hand, the AA's prediction will be close to 0, and so the cumulative loss of the AA over the first $T$ trials will be about $T$, which is close to the best expert's loss. We can see that the ridge regression procedure in this situation is forced to suffer a loss 4 times as big as the AA's loss.

The lower bound stated in Theorem 2 does not imply that our regression algorithm is better than the ridge regression procedure in our adversarial framework. (Moreover, the idea of our proof of Theorem 2 is to lower bound the performance of the ridge regression procedure in the situation where the expected loss of the ridge regression procedure is optimal.) Theorem 1 asserts that

$$L_T(\text{Learner}) \le \inf_w \left( L_T(w) + a\|w\|^2 \right) + Y^2 \sum_{i=1}^{n} \ln \left( 1 + \frac{1}{a} \sum_{t=1}^{T} x_{t,i}^2 \right) \tag{5}$$

when Learner follows the AA. The next theorem shows that the ridge regression procedure sometimes violates this inequality.

**Theorem 3** *Let $n = 1$ (the number of attributes) and $Y = 1$ (the upper bound on $|y_t|$); fix $a > 0$. Nature has a strategy such that, when Learner plays the ridge regression strategy,*

$$L_T(\text{Learner}) = 4T + O(1), \tag{6}$$

$$\inf_w \left( L_T(w) + a\|w\|^2 \right) = T + O(1), \tag{7}$$

$$\ln \left( 1 + \frac{1}{a} \sum_{t=1}^{T} x_t^2 \right) = T \ln 2 + O(1) \tag{8}$$

*as $T \to \infty$ (and, therefore, (5) is violated).*

## 5 CONCLUSION

A distinctive feature of our approach to linear regression is that our only assumption about the data is that $|y_t| \le Y$, $\forall t$; we do not make any assumptions about stochastic properties of the data-generating mechanism. In some situations (if the data were generated by a partially known stochastic mechanism) this feature is a disadvantage, but often it will be an advantage.

This paper was greatly influenced by Vapnik's [8] idea of transductive inference. The algorithm analyzed in this paper is "transductive", in the sense that it outputs some prediction $p_t$ for $y_t$ after being given $x_t$, rather than to output a general rule for mapping $x_t$ into $p_t$; in particular, $p_t$ may depend non-linearly on $x_t$. (It is easy, however, to extract such a rule from the description of the algorithm once it is found.)

**Acknowledgments**

Kostas Skouras and Philip Dawid noticed that our regression algorithm is different from the ridge regression and that in some situations it behaves very differently. Manfred Warmuth's advice about relevant literature is also gratefully appreciated.

# References

[1] N. Cesa-Bianchi, P. M. Long, and M. K. Warmuth (1996), Worst-case quadratic loss bounds for on-line prediction of linear functions by gradient descent, *IEEE Trans. Neural Networks* **7**:604–619.

[2] A. DeSantis, G. Markowsky, and M. N. Wegman (1988), Learning probabilistic prediction functions, *in* "Proceedings, 29th Annual IEEE Symposium on Foundations of Computer Science," pp. 110–119, Los Alamitos, CA: IEEE Comput. Soc.

[3] D. P. Foster (1991), Prediction in the worst case, *Ann. Statist.* **19**:1084–1090.

[4] Y. Freund (1996), Predicting a binary sequence almost as well as the optimal biased coin, *in* "Proceedings, 9th Annual ACM Conference on Computational Learning Theory", pp. 89–98, New York: Assoc. Comput. Mach.

[5] D. Haussler, J. Kivinen, and M. K. Warmuth (1994), Tight worst-case loss bounds for predicting with expert advice, University of California at Santa Cruz, Technical Report UCSC-CRL-94-36, revised December. Short version *in* "Computational Learning Theory" (P. Vitányi, Ed.), Lecture Notes in Computer Science, Vol. 904, pp. 69–83, Berlin: Springer, 1995.

[6] J. Kivinen and M. K. Warmuth (1997), Exponential Gradient versus Gradient Descent for linear predictors, *Inform. Computation* **132**:1–63.

[7] N. Littlestone and M. K. Warmuth (1994), The Weighted Majority Algorithm, *Inform. Computation* **108**:212–261.

[8] V. N. Vapnik (1995), *The Nature of Statistical Learning Theory*, New York: Springer.

[9] V. Vovk (1990), Aggregating strategies, *in* "Proceedings, 3rd Annual Workshop on Computational Learning Theory" (M. Fulk and J. Case, Eds.), pp. 371–383, San Mateo, CA: Morgan Kaufmann.

[10] V. Vovk (1992), Universal forecasting algorithms, *Inform. Computation* **96**:245–277.

[11] V. Vovk (1997), A game of prediction with expert advice, to appear in *J. Comput. Inform. Syst.* Short version *in* "Proceedings, 8th Annual ACM Conference on Computational Learning Theory," pp. 51–60, New York: Assoc. Comput. Mach., 1995.

[12] K. Yamanishi (1997), A decision-theoretic extension of stochastic complexity and its applications to learning, submitted to *IEEE Trans. Inform. Theory.*